# On the Distribution of the Number of Local Minima of a Random Function on a Graph

**Pierre Baldi**
JPL, Caltech
Pasadena, CA 91109

**Yosef Rinott**
UCSD
La Jolla, CA 92093

**Charles Stein**
Stanford University
Stanford, CA 94305

## 1    INTRODUCTION

Minimization of energy or error functions has proved to be a useful principle in the design and analysis of neural networks and neural algorithms. A brief list of examples include: the back- propagation algorithm, the use of optimization methods in computational vision, the application of analog networks to the approximate solution of NP complete problems and the Hopfield model of associative memory.

In the Hopfield model associative memory, for instance, a quadratic Hamiltonian of the form

$$F(x) = \frac{1}{2} \sum_{i,j=1}^{n} w_{ij} x_i x_j \qquad x_i = \pm 1 \tag{1}$$

is constructed to tailor a particular "landscape" on the $n$- dimensional hypercube $H^n = \{-1, 1\}^n$ and store memories at a particular subset of the local minima of $F$ on $H^n$. The synaptic weights $w_{ij}$ are usually constructed incrementally, using a form of Hebb's rule applied to the patterns to be stored. These patterns are often chosen at random. As the number of stored memories grows to and beyond saturation, the energy function $F$ becomes essentially random. In addition, in a general context of combinatorial optimization, every problem in NP can be (polynomially) reduced to the problem of minimizing a certain quadratic form over $H^n$.

These two types of considerations, associative memory and combinatorial optimization, motivate the study of the number and distribution of local minima of a random function $F$ defined over the hypercube, or more generally, any graph $G$. Of course, different notions of randomness can be introduced. In the case where $F$ is a

quadratic form as in (1), we could take the coefficients $w_{ij}$ to be independent identically distributed gaussian random variables, which yields, in fact, the Sherrington-Kirkpatrick long-range spin glass model of statistical physics. For this model, the expectation of the number of local minima is well known but no rigorous results have been obtained for its distribution (even the variance is not known precisely). A simpler model of randomness can then be introduced, where the values $F(x)$ of the random function at each vertex are assigned randomly and independently from a common distribution: This is in fact the random energy model of Derrida (1981).

## 2    THE MAIN RESULT

In Baldi, Rinott and Stein (1989) the following general result on random energy models is proven.

Let $G = (V, E)$ be a regular $d$-graph, i.e., a graph where every vertex has the same number $d$ of neighbors. Let $F$ be a random function on $V$ whose values are independently distributed with a common continuous distribution. Let $W$ be the number of local minima of $F$, i.e., the number of vertices $x$ satisfying $F(x) > F(y)$ for any neighbor $y$ of $x$ (i.e., $(x,y)\epsilon E$). Let $EW = \lambda$ and Var $W = \sigma^2$. Then

$$EW = \frac{|V|}{d+1} \tag{2}$$

and for any positive real $w$:

$$\left| P(W \le w) - \Phi\left(\frac{w-\lambda}{\sigma}\right) \right| \le \frac{C}{\sqrt{\sigma}} \tag{3}$$

where $\Phi$ is the standard normal distribution and $C$ is an absolute constant.

Remarks:

(a) The proof of (3) ((2) is obvious) is based on a method developed in Stein (1986).

(b) The bound given in the theorem is not asymptotic but holds also for small graphs.

(c) If $|V| \rightarrow \infty$ the theorem states that if $\sigma \rightarrow \infty$ then the distribution of the number of local minima approaches a normal distribution and (3) gives also a bound of $O(\sigma^{-1/2})$ on the rate of convergence.

(d) The function $F$ simply induces a ranking (or a random permutation) of the vertices of $G$.

(e) The bound in (3) may not be optimal. We suspect that the optimal rate should scale like $\sigma^{-1}$ rather than $\sigma^{-1/2}$.

## 3   EXAMPLES OF APPLICATIONS

(1) Consider a $n \times n$ square lattice (see fig.1) with periodic boundary conditions. Here, $|V_n| = n^2$ and $d = 4$. The expected number of local minima is

$$EW_n = \frac{n^2}{5} \tag{4}$$

and a simple calculations shows that

$$\mathrm{Var}\, W_n = \frac{13n^2}{225}. \tag{5}$$

Therefore $W_n$ is asymptotically normal and the rate of convergence is bounded by $O(n^{-1/2})$.

(2) Consider a $n \times n$ square lattice, where this time the neighbors of a vertex $v$ are all the points in same row or column as $v$ (see fig.2). This example arises in game theory, where the rows (resp. columns) correspond to different possible strategies of one of two players. The energy value can be interpreted as the cost of the combined choice of two strategies. Here $|V_n| = n^2$ and $d = 2n - 2$. The expected number of local minima (the Nash equilibrium points of game theory) $W_n$ is

$$EW_n = \frac{n^2}{2n-1} \approx \frac{n}{2} \tag{6}$$

and

$$\mathrm{Var}\, W_n = \frac{n^2(n-1)}{2(2n-1)^2} \approx \frac{n}{8}. \tag{7}$$

Therefore $W_n$ is asymptotically normal and the rate of convergence is bounded by $O(n^{-1/4})$.

(3) Consider the $n$-dimensional hypercube $H^n = (V_n, E_n)$ (see fig.3). Then $|V_n| = 2^n$ and $d = n$. The expected number of local minima $W_n$ is:

$$EW_n = \frac{2^n}{n+1} = \lambda_n \tag{8}$$

and

$$\mathrm{Var}\, W_n = \frac{2^{n-1}(n-1)}{(n+1)^2} = \sigma_n^2. \tag{9}$$

Therefore $W_n$ is asymptotically normal and in fact:

$$\left| P(w_n \leq w) - \Phi\left(\frac{w - \lambda_n}{\sigma_n}\right) \right| \leq \frac{C\sqrt{n+1}}{(n-1)^{1/4}2^{(n-1)/4}} = O(\sqrt[4]{n/2^n}). \tag{10}$$

In contrast, if the edges of $H^n$ are randomly and independently oriented with probability .5, then the distribution of the number of vertices having all their adjacent edges oriented inward is asymptotically Poisson with mean 1.

## References

P. Baldi, Y. Rinott (1989), "Asymptotic Normality of Some Graph-Related Statistics," Journal of Applied Probability, 26, 171-175.

P. Baldi and Y. Rinott (1989), "On Normal Approximation of Distribution in Terms of Dependency Graphs," Annals of Probability, in press.

P. Baldi, Y. Rinott and C. Stein (1989), "A Normal Approximation for the Number of Local Maxima of a Random Function on a Graph," In: Probability, Statistics and Mathematics: Papers in Honor of Samuel Karlin. T.W. Anderson, K.B. Athreya and D.L. Iglehard, Editors, Academic Press.

B. Derrida (1981), "Random Energy Model: An Exactly Solvable Model of Disordered Systems," Physics Review, B24, 2613- 2626.

C. M. Macken and A. S. Perelson (1989), "Protein Evolution on Rugged Landscapes", PNAS, 86, 6191-6195.

C. Stein (1986), "Approximate Computation of Expectations," Institute of Mathematical Statistics Lecture Notes, S.S. Gupta Series Editor, Volume 7.

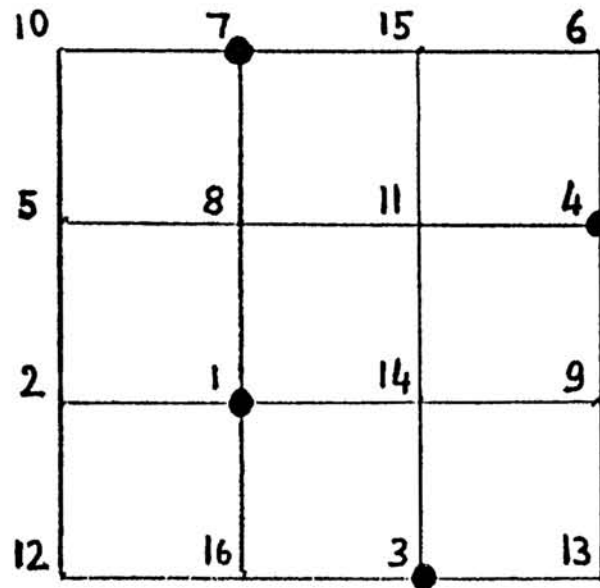

**Figure 1:**       A ranking of a 4 × 4 square lattice with periodic boundary conditions and four local minima ($d = 4$).

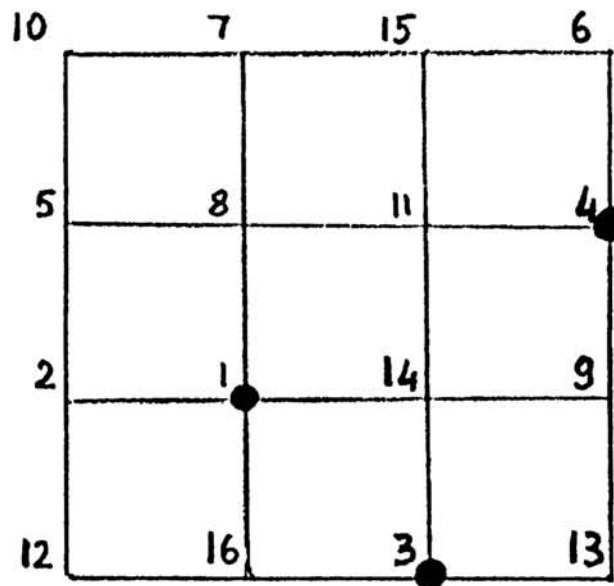

**Figure 2:**       A ranking of a 4 × 4 square lattice. The neighbors of a vertex are all the points on the same row and column. There are three local minima ($d = 6$).

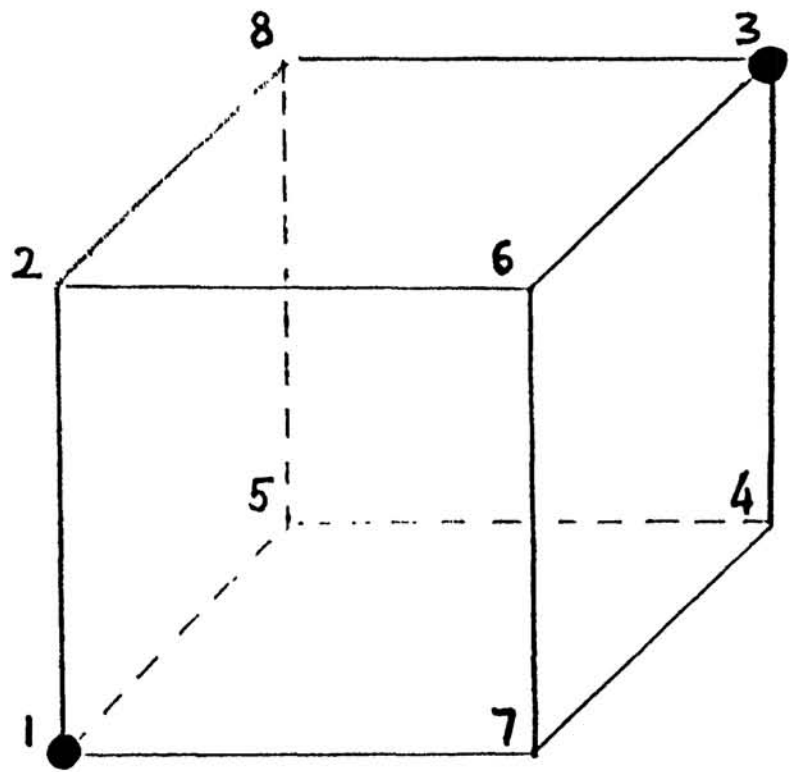

**Figure 3:**     A ranking of $H^3$ with two local minima ($d = 3$).